# A generative model for attractor dynamics

**Richard S. Zemel**
Department of Psychology
University of Arizona
Tucson, AZ 85721
zemel@u.arizona.edu

**Michael C. Mozer**
Department of Computer Science
University of Colorado
Boulder, CO 80309-0430
mozer@colorado.edu

## Abstract

Attractor networks, which map an input space to a discrete output space, are useful for pattern completion. However, designing a net to have a given set of attractors is notoriously tricky; training procedures are CPU intensive and often produce spurious attractors and ill-conditioned attractor basins. These difficulties occur because each connection in the network participates in the encoding of multiple attractors. We describe an alternative formulation of attractor networks in which the encoding of knowledge is local, not distributed. Although localist attractor networks have similar dynamics to their distributed counterparts, they are much easier to work with and interpret. We propose a statistical formulation of localist attractor net dynamics, which yields a convergence proof and a mathematical interpretation of model parameters.

Attractor networks map an input space, usually continuous, to a sparse output space composed of a discrete set of alternatives. Attractor networks have a long history in neural network research.

Attractor networks are often used for *pattern completion*, which involves filling in missing, noisy, or incorrect features in an input pattern. The initial state of the attractor net is typically determined by the input pattern. Over time, the state is drawn to one of a predefined set of states—the *attractors*. Attractor net dynamics can be described by a state trajectory (Figure 1a). An attractor net is generally implemented by a set of visible units whose activity represents the instantaneous state, and optionally, a set of hidden units that assist in the computation. Attractor dynamics arise from interactions among the units. In most formulations of attractor nets,[2,3] the dynamics can be characterized by gradient descent in an energy landscape, allowing one to partition the output space into *attractor basins*. Instead of homogeneous attractor basins, it is often desirable to sculpt basins that depend on the recent history of the network and the arrangement of attractors in the space. In psychological models of human cognition, for example, *priming* is fundamental: after the model visits an attractor, it should be faster to fall into the same attractor in the near future, i.e., the attractor basin should be broadened.[1,6]

Another property of attractor nets is key to explaining behavioral data in psychological and neurobiological models: the *gang effect*, in which the strength of an attractor is influenced by other attractors in its neighborhood. Figure 1b illustrates the gang effect: the proximity of the two rightmost attractors creates a deeper attractor basin, so that if the input starts at the origin it will get pulled to the right.

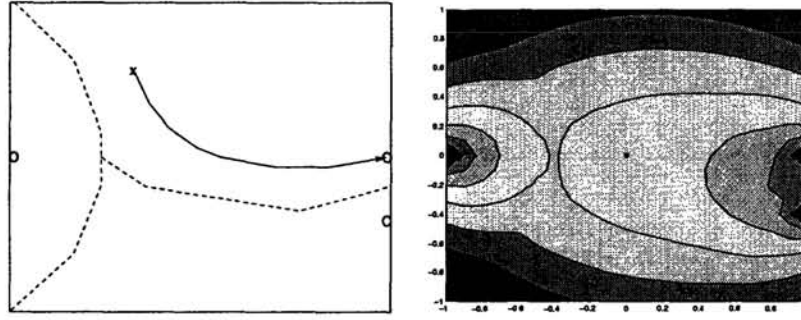

Figure 1: (a) A two-dimensional space can be carved into three regions (dashed lines) by an attractor net. The dynamics of the net cause an input pattern (the X) to be mapped to one of the attractors (the O's). The solid line shows the temporal trajectory of the network state. (b) the actual energy landscape for a localist attractor net as a function of $\hat{y}$, when the input is fixed at the origin and there are three attractors, $\mathbf{w} = ((-1, 0), (1, 0), (1, -.4))$, with a uniform prior. The shapes of attractor basins are influenced by the proximity of attractors to one another (the *gang effect*). The origin of the space (depicted by a point) is equidistant from the attractor on the left and the attractor on the upper right, yet the origin clearly lies in the basin of the right attractors.

This effect is an emergent property of the distribution of attractors, and is the basis for interesting dynamics; it produces the mutually reinforcing or inhibitory influence of similar items in domains such as semantics,[9] memory,[10,12] and olfaction.[4]

Training an attractor net is notoriously tricky. Training procedures are CPU intensive and often produce spurious attractors and ill-conditioned attractor basins.[5,11] Indeed, we are aware of no existing procedure that can robustly translate an arbitrary specification of an attractor landscape into a set of weights. These difficulties are due to the fact that each connection participates in the specification of multiple attractors; thus, knowledge in the net is *distributed* over connections.

We describe an alternative attractor network model in which knowledge is *localized*, hence the name *localist attractor network*. The model has many virtues, including: a trivial procedure for wiring up the architecture given an attractor landscape; eliminating spurious attractors; achieving gang effects; providing a clear mathematical interpretation of the model parameters, which clarifies how the parameters control the qualitative behavior of the model (e.g., the magnitude of gang effects); and proofs of convergence and stability.

A localist attractor net consists of a set of $n$ *state units* and $m$ *attractor units*. Parameters associated with an attractor unit $i$ encode the location of the attractor, denoted $\mathbf{w}_i$, and its "pull" or strength, denoted $\pi_i$, which influence the shape of the attractor basin. Its activity at time $t$, $q_i(t)$, reflects the normalized distance from the attractor center to the current state, $\mathbf{y}(t)$, weighted by the attractor strength:

$$q_i(t) = \frac{\pi_i g(\mathbf{y}(t), \mathbf{w}_i, \sigma(t))}{\sum_j \pi_j g(\mathbf{y}(t), \mathbf{w}_j, \sigma(t))} \tag{1}$$

$$g(\mathbf{y}, \mathbf{w}, \sigma) = \exp(-|\mathbf{y} - \mathbf{w}|^2 / 2\sigma^2) \tag{2}$$

Thus, the attractors form a layer of normalized radial-basis-function units.

The input to the net, $\mathcal{E}$, serves as the initial value of the state, and thereafter the state is pulled toward attractors in proportion to their activity. A straightforward

expression of this behavior is:

$$\mathbf{y}(t+1) = \alpha(t)\mathcal{E} + (1 - \alpha(t)) \sum_i q_i(t)\mathbf{w}_i. \tag{3}$$

where $\alpha(1) = 1$ on the first update and $\alpha(t) = 0$ for $t > 1$. More generally, however, one might want to gradually reduce $\alpha$ over time, allowing for a persistent effect of the external input on the asymptotic state. The variables $\sigma(t)$ and $\alpha(t)$ are not free parameters of the model, but can be derived from the formalism we present below.

The localist attractor net is motivated by a generative model of the input based on the attractor distribution, and the network dynamics corresponds to a search for a maximum likelihood interpretation of the observation. In the following section, we derive this result, and then present simulation studies of the architecture.

## 1  A MAXIMUM LIKELIHOOD FORMULATION

The starting point for the statistical formulation of a localist attractor network is a mixture of Gaussians model. A standard mixture of Gaussians consists of $m$ Gaussian density functions in $n$ dimensions. Each Gaussian is parameterized by a mean, a covariance matrix, and a mixture coefficient. The mixture model is *generative*, i.e., it is considered to have produced a set of observations. Each observation is generated by selecting a Gaussian based on the mixture coefficients and then stochastically selecting a point from the corresponding density function. The model parameters are adjusted to maximize the likelihood of a set of observations. The Expectation-Maximization (EM) algorithm provides an efficient procedure for estimating the parameters.The Expectation step calculates the posterior probability $q_i$ of each Gaussian for each observation, and the Maximization step calculates the new parameters based on the previous values and the set of $q_i$.

The mixture of Gaussians model can provide an interpretation for a localist attractor network, in an unorthodox way. Each Gaussian corresponds to an attractor, and an observation corresponds to the state. Now, however, instead of fixing the observation and adjusting the Gaussians, we fix the Gaussians and adjust the observation. If there is a single observation, and $\alpha = 0$ and all Gaussians have uniform spread $\sigma$, then Equation 1 corresponds to the Expectation step, and Equation 3 to the Maximization step in this unusual mixture model.

Unfortunately, this simple characterization of the localist attractor network does not produce the desired behavior. Many situations produce partial solutions, in which the observation does not end up at an attractor. For example, if two unidimensional Gaussians overlap significantly, the most likely value for the observation is midway between them rather than at the mean of either Gaussian.

We therefore extend this mixture-of-Gaussians formulation to better characterize the localist attractor network. As in the simple model, each of the $m$ attractors is a Gaussian generator, the mean of which is a location in the $n$-dimensional state space. The input to the net, $\mathcal{E}$, is considered to have been generated by a stochastic selection of one of the attractors, followed by the addition of zero-mean Gaussian noise with variance specified by the attractor. Given a particular observation $\mathcal{E}$, the an attractor's posterior probability is the normalized Gaussian probability of $\mathcal{E}$, weighted by its mixing proportion. This posterior distribution for the attractors corresponds to a distribution in state space that is a weighted sum of Gaussians.

We then consider the attractor network as encoding this distribution over states implied by the attractor posterior probabilities. At any one time, however, the attractor network can only represent a single position in state space, rather than

the entire distribution over states. This restriction is appropriate when the state is an $n$-dimensional point represented by the pattern of activity over $n$ state units. To accommodate this restriction, we change the standard mixture of Gaussians generative model by interjecting an intermediate level between the attractors and the observation. The first generative level consists of the discrete attractors, the second is the state space, and the third is the observation. Each observation is considered to have been generated by moving down this hierarchy:

1. select an attractor $\mathbf{x} = i$ from the set of attractors

2. select a state (i.e., a pattern of activity across the state units) based on the preferred location of that attractor: $\mathbf{y} = \mathbf{w}_i + \mathcal{N}_y$

3. select an observation $\mathbf{z} = \mathbf{y}\mathbf{G} + \mathcal{N}_z$

The observation $\mathbf{z}$ produced by a particular state $\mathbf{y}$ depends on the generative weight matrix $\mathbf{G}$. In the networks we consider here, the observation and state spaces are identical, so $\mathbf{G}$ is the identity matrix, but the formulation allows for $\mathbf{z}$ to lie in some other space. $\mathcal{N}_y$ and $\mathcal{N}_z$ describe the zero-mean, spherical Gaussian noise introduced at the two levels, with deviations $\sigma_y$ and $\sigma_z$, respectively.

In comparison with the 2-level Gaussian mixture model described above, this 3-level model is more complicated but more standard: the observation $\mathcal{E}$ is preserved as stable data, and rather than the model manipulating the data here it can be viewed as iteratively manipulating an internal representation that fits the observation and attractor structure. The attractor dynamics correspond to an iterative search through state space to find the most likely single state that: (a) was generated by the mixture of Gaussian attractors, and (b) in turn generated the observation.

Under this model, one could fit an observation $\mathcal{E}$ by finding the posterior distribution over the hidden states ($X$ and $\mathbf{Y}$) given the observation:

$$p(X = i, \mathbf{Y} = \mathbf{y} | \mathbf{Z} = \mathcal{E}) = \frac{p(\mathcal{E}|\mathbf{y}, i)p(\mathbf{y}, i)}{p(\mathcal{E})} = \frac{p(\mathcal{E}|\mathbf{y})\pi_i p(\mathbf{y}|i)}{\int_{\mathbf{y}} p(\mathcal{E}|\mathbf{y})\sum_i \pi_i p(\mathbf{y}|i)d\mathbf{y}} \quad (4)$$

where the conditional distributions are Gaussian: $p(\mathbf{Y} = \mathbf{y}|X = i) = \mathcal{G}(\mathbf{y}|\mathbf{w}_i, \sigma_y)$ and $p(\mathcal{E}|\mathbf{Y} = \mathbf{y}) = \mathcal{G}(\mathcal{E}|\mathbf{y}, \sigma_z)$. Evaluating the distribution in Equation 4 is tractable, because the partition function is a sum of a set of Gaussian integrals. Due to the restriction that the network cannot represent the entire distribution, we do not directly evaluate this distribution but instead adopt a mean-field approach, in which we approximate the posterior by another distribution $Q(X, \mathbf{Y}|\mathcal{E})$. Based on this approximation, the network dynamics can be seen as minimizing an objective function that describes an upper bound on the negative log probability of the observation given the model and mean-field parameters.

In this approach, one can choose any form of $Q$ to estimate the posterior distribution, but a better estimate allows the network to approach a maximum likelihood solution.[13] We select a simple posterior: $Q(X, \mathbf{Y}) = q_i \delta(\mathbf{Y} = \hat{\mathbf{y}})$, where $q_i = Q(X = i)$ is the responsibility assigned to attractor $i$, and $\hat{\mathbf{y}}$ is the estimate of the state that accounts for the observation. The delta function over $\mathbf{Y}$ is motivated by the restriction that the explanation of an input consists of a single state.

Given this posterior distribution, the objective for the network is to minimize the free energy $F$, described here for a particular input example $\mathcal{E}$:

$$
\begin{aligned}
F(\mathbf{q}, \hat{y}|\mathcal{E}) &= \sum_i \int Q(X = i, \mathbf{Y} = \mathbf{y}) \ln \frac{Q(X = i, \mathbf{Y} = \mathbf{y})}{P(\mathcal{E}, X = i, \mathbf{Y} = \mathbf{y})} d\mathbf{y} \\
&= \sum_i q_i \ln \frac{q_i}{\pi_i} - \ln p(\mathcal{E}|\hat{y}) - \sum_i q_i \ln p(\hat{y}|i)
\end{aligned}
$$

where $\pi_i$ is the prior probability (mixture coefficient) associated with attractor $i$. These priors are parameters of the generative model, as are $\sigma_y$, $\sigma_z$, and $\mathbf{w}$.

$$F(\mathbf{q}, \hat{\mathbf{y}}|\mathcal{E}) = \sum_i q_i \ln \frac{q_i}{\pi_i} + \frac{1}{2\sigma_z^2}|\mathcal{E} - \hat{\mathbf{y}}|^2 + \frac{1}{2\sigma_y^2}\sum_i q_i|\hat{\mathbf{y}} - \mathbf{w}_i|^2 + n\ln(\sigma_y\sigma_z) \quad (5)$$

Given an observation, a good set of mean-field parameters can be determined by alternating between updating the generative parameters and the mean-field parameters. The update procedure is guaranteed to converge to a minimum of $F$, as long as the updates are done asynchronously and each update minimizes $F$ with respect to a parameter.[8] The update equations for the mean-field parameters are:

$$\hat{\mathbf{y}} = \frac{\sigma_y^2\mathcal{E} + \sigma_z^2\sum_i q_i\mathbf{w}_i}{\sigma_y^2 + \sigma_z^2} \quad (6)$$

$$q_i = \frac{\pi_i p(\hat{\mathbf{y}}|i)}{\sum_j \pi_j p(\hat{\mathbf{y}}|j)} \quad (7)$$

In our simulations, we hold most of the parameters of the generative model constant, such as the priors $\pi$, the weights $\mathbf{w}$, and the generative noise in the observation, $\sigma_z$. The only aspect that changes is the generative noise in the state, $\sigma_y$, which is a single parameter shared by all attractors:

$$\sigma_y^2 = \frac{1}{d}\sum_i q_i|\hat{\mathbf{y}} - \mathbf{w}_i|^2 \quad (8)$$

The updates of Equations 6-8 can be in any order. We typically initialize the state $\hat{\mathbf{y}}$ to $\mathcal{E}$ at time 0, and then cyclically update the $q_i$, $\sigma_y$, then $\hat{\mathbf{y}}$.

This generative model avoids the problem of spurious attractors described above for the standard Gaussian mixture model. Intuition into how the model avoids spurious attractors can be gained by inspecting the update equations. These equations effectively tie together two processes: moving $\hat{\mathbf{y}}$ closer to some $\mathbf{w}_i$ than the others, and increasing the corresponding responsibility $q_i$. As these two processes evolve together, they act to descrease the noise $\sigma_y$, which accentuates the pull of the attractor. Thus stable points that do not correspond to the attractors are rare.

## 2 SIMULATION STUDIES

To create an attractor net, we specify the parameters $(\pi_i, \mathbf{w}_i)$ associated with the attractors based on the desired structure of the energy landscape (e.g., Figure 1b). The only remaining free parameter, $\sigma_z$, plays an important role in determining how responsive the system is to the external input.

We have conducted several simulation studies to explore properties of localist attractor networks. Systematic investigations with a 200-dimensional state space and 200 attractors, randomly placed at corners of the 200-D hypercube, have demonstrated that spurious responses are exceedingly rare unless more than 85% of an input's features are distorted (Figure 2), and that manipulating parameters such as noise and prior probabilities has the predicted effects. We have also conducted studies of localist attractor networks in the domain of visual images of faces. These simulations have shown that gang effects arise when there is structure among the attractors. For example, when the attractor set consists of a single view of several different faces, and multiple views of one face, then an input that is a morphed face—a linear combination of one of the single-view faces and one view of the gang face—will end up in the gang attractor even when the initial weighting assigned to the gang face was less than 40%.

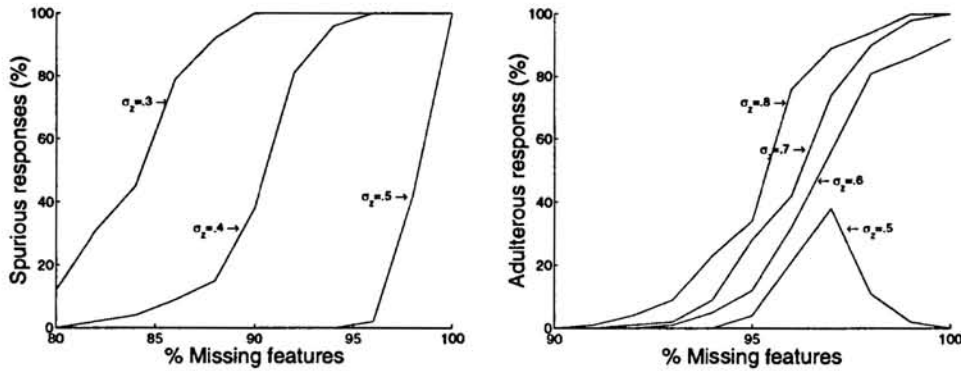

Figure 2: The input must be severely corrupted before the net makes *spurious* (final state not at an attractor) or *adulterous* (final state at a neighbor of the generating attractor) responses. (a) The percentage of spurious responses increases as $\sigma_z$ is increased. (b) The percentage of adulterous responses increases as $\sigma_z$ is decreased.

To test the architecture on a larger, structured problem, we modeled the domain of three-letter English words. The idea is to use the attractor network as a content addressable memory which might, for example, be queried to retrieve a word with P in the third position and any letter but A in the second position, a word such as HIP. The attractors consist of the 423 three-letter English words, from ACE to ZOO. The state space of the attractor network has one dimension for each of the 26 letters of the English alphabet in each of the 3 positions, for a total of 78 dimensions. We can refer to a given dimension by the letter and position it encodes, e.g., $P_3$ denotes the dimension corresponding to the letter P in the third position of the word. The attractors are at the corners of a $[-1, +1]^{78}$ hypercube. The attractor for a word such as HIP is located at the state having value $-1$ on all dimensions except for $H_1$, $I_2$, and $P_3$ which have value $+1$. The external input specifies a state that constrains the solution. For example, one might specify "P in the third position" by setting the external input to $+1$ on dimension $P_3$ and to $-1$ on dimensions $\alpha_3$, for all letters $\alpha$ other than P. One might specify the absence of a constraint in a particular letter position, $\rho$, by setting the external input to 0 on dimensions $\alpha_\rho$, for all letters $\alpha$.

The network's task is to settle on a state corresponding to one of the words, given soft constraints on the letters. The interactive-activation model of word perception[7] performs a similar computation, and our implementation exhibits the key qualitative properties of their model. If the external input specifies a word, of course the attractor net will select that word. Interesting queries are those in which the external input underconstrains or overconstrains the solution. We illustrate with one example of the network's behavior, in which the external input specifies $D_1$, $E_2$, and $G_3$. Because DEG is a nonword, no attractor exists for that state. The closest attractors share two letters with DEG, e.g., PEG, BEG, DEN, and DOG. Figure 3 shows the effect of gangs on the selection of a response, BEG.

## 3   CONCLUSION

Localist attractor networks offer an attractive alternative to standard attractor networks, in that their dynamics are easy to control and adapt. We described a statistical formulation of a type of localist attractor, and showed that it provides a Lyapunov function for the system as well as a mathematical interpretation for the network parameters. The dynamics of this system are derived not from intuitive arguments but from this formal mathematical model. Simulation studies show that the architecture achieves gang effects, and spurious attractors are rare. This approach is inefficient if the attractors have compositional structure, but for many applications of pattern recognition or associative memory, the number of items

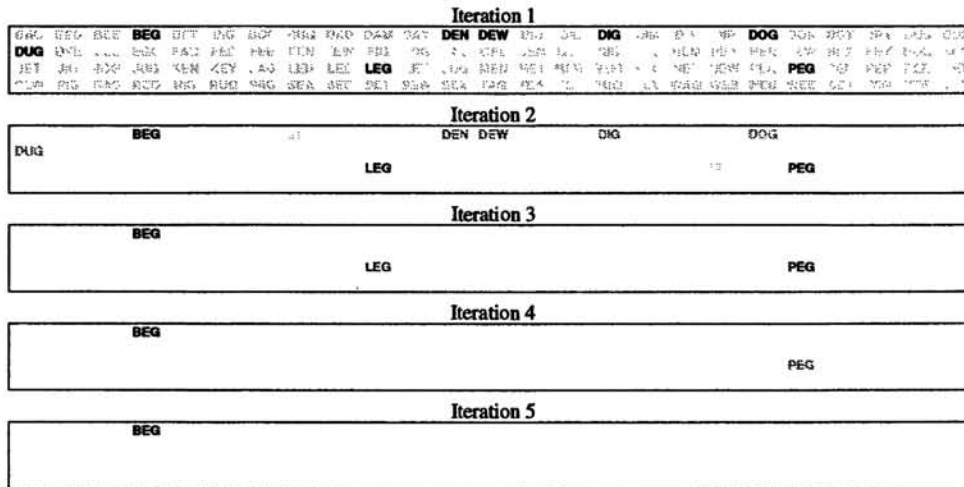

Figure 3: Simulation of the 3-letter word attractor network, queried with DEG. Each frame shows the relative activity of attractor units at various points in processing. Activity in each frame is normalized such that the most active unit is printed in black ink; the lighter the ink color, the less active the unit. Only attractor units sharing at least one letter with DEG are shown. The selection, BEG, is a product of a gang effect. The gangs in this example are formed by words sharing two letters. The most common word beginnings are PE– (7 instances) and DI– (6); the most common word endings are –AG (10) and –ET (10); the most common first-last pairings are B–G (5) and D–G (3). One of these gangs supports $B_1$, two support $E_2$, and three support $G_3$, hence BEG is selected.

being stored is small. The approach is especially useful in cases where attractor locations are known, and the key focus of the network is the mutual influence of the attractors, as in many cognitive modelling studies.

# References

[1] Becker, S., Moscovitch, M., Behrmann, M., & Joordens, S. (1997). Long-term semantic priming: A computational account and empirical evidence. *Journal of Experimental Psychology: Learning, Memory, & Cognition, 23(5)*, 1059-1082.

[2] Golden, R. (1988). Probabilistic characterization of neural model computations. In D. Z. Anderson (Ed.), *Neural Information Processing Systems* (pp. 310-316). American Institute of Physics.

[3] Hopfield, J. J. (1982). Neural networks and physical systems with emergent collective computational abilities. *Proceedings of the National Academy of Sciences, 79*, 2554-2558.

[4] Kay, L.M., Lancaster, L.R., & Freeman W.J. (1996). Reafference and attractors in the olfactory system during odor recognition. *Int J Neural Systems, 7(4)*, 489-95.

[5] Mathis, D. (1997). *A computational theory of consciousness in cognition*. Unpublished Doctoral Dissertation. Boulder, CO: Department of Computer Science, University of Colorado.

[6] Mathis, D., & Mozer, M. C. (1996). Conscious and unconscious perception: A computational theory. In G. Cottrell (Ed.), *Proceedings of the Eighteenth Annual Conference of the Cognitive Science Society* (pp. 324-328). Erlbaum.

[7] McClelland, J. L. & Rumelhart, D. E. (1981). An interactive activation model of context effects in letter perception: Part I. An account of basic findings. *Psychological Review, 88*, 375-407.

[8] Neal, R. M. & Hinton, G. E. (1998). A view of the EM algorithm that justifies incremental, sparse, and other variants. In M. I. Jordan (Ed.), *Learning in Graphical Models*. Kluwer Academic Press.

[9] McRae, K., de Sa, V. R., & Seidenberg, M. S. (1997) On the nature and scope of featural representations of word meaning. *Journal of Experimental Psychology: General, 126(2)*, 99-130.

[10] Redish, A. D. & Touretzky, D. S. (1998). The role of the hippocampus in solving the Morris water maze. *Neural Computation, 10(1)*, 73-111.

[11] Rodrigues, N. C., & Fontanari, J. F. (1997). Multivalley structure of attractor neural networks. *Journal of Physics A (Mathematical and General), 30*, 7945-7951.

[12] Samsonovich, A. & McNaughton, B. L. (1997) Path integration and cognitive mapping in a continuous attractor neural network model. *Journal of Neuroscience, 17(15)*, 5900-5920.

[13] Saul, L.K., Jaakkola, T., & Jordan, M.I. (1996). Mean field theory for sigmoid belief networks. *Journal of AI Research, 4*, 61–76.
